# On the Complexity of Learning the Kernel Matrix

**Olivier Bousquet,  Daniel J. L. Herrmann**
MPI for Biological Cybernetics
Spemannstr. 38, 72076 Tübingen
Germany
{*olivier.bousquet, daniel.herrmann*}*@tuebingen.mpg.de*

## Abstract

We investigate data based procedures for selecting the kernel when learning with Support Vector Machines. We provide generalization error bounds by estimating the Rademacher complexities of the corresponding function classes. In particular we obtain a complexity bound for function classes induced by kernels with given eigenvectors, i.e., we allow to vary the spectrum and keep the eigenvectors fix. This bound is only a logarithmic factor bigger than the complexity of the function class induced by a single kernel. However, optimizing the margin over such classes leads to overfitting. We thus propose a suitable way of constraining the class. We use an efficient algorithm to solve the resulting optimization problem, present preliminary experimental results, and compare them to an alignment-based approach.

## 1   Introduction

Ever since the introduction of the Support Vector Machine (SVM) algorithm, the question of choosing the kernel has been considered as crucial. Indeed, the success of SVM can be attributed to the joint use of a robust classification procedure (large margin hyperplane) and of a convenient and versatile way of pre-processing the data (kernels). It turns out that with such a decomposition of the learning process into preprocessing and linear classification, the performance highly depends on the preprocessing and much less on the linear classification algorithm to be used (e.g. the kernel perceptron has been shown to have comparable performance to SVM with the same kernel). It is thus of high importance to have a criterion to choose the suitable kernel for a given problem.

Ideally, this choice should be dictated by the data itself and the kernel should be 'learned' from the data. The simplest way of doing so is to choose a parametric family of kernels (such as polynomial or Gaussian) and to choose the values of the parameters by cross-validation. However this approach is clearly limited to a small number of parameters and requires the use of extra data.

Chapelle et al. [1] proposed a different approach. They used a bound on the generalization error and computed the gradient of this bound with respect to the kernel parameters. This allows to perform a gradient descent optimization and thus to effectively handle a large number of parameters.

More recently, the idea of using non-parametric classes of kernels has been proposed by Cristianini et al. [2]. They work in a transduction setting where the test data is known in advance. In that setting, the kernel reduces to a positive definite matrix of fixed size (Gram matrix). They consider the set of kernel matrices with given eigenvectors and to choose the eigenvalues using the 'alignment' between the kernel and the data. This criterion has the advantage of being easily computed and optimized. However it has no direct connection to the generalization error.

Lanckriet et al. [5] derived a generalization bound in the transduction setting and proposed to use this bound to choose the kernel. Their parameterization is based on a linear combination of given kernel matrices and their bound has the advantage of leading to a convex criterion. They thus proposed to use semidefinite programming for performing the optimization.

Actually, if one wants to have a feasible optimization, one needs the criterion to be *nice* (e.g. differentiable) and the parameterization to be *nice* (e.g. the criterion is convex with respect to the parameters). The criterion and parameterization proposed by Lanckriet et al. satisfy these requirements. We shall use their approach and develop it further.

In this paper, we try to combine the advantages of previous approaches. In particular we propose several classes of kernels and give bounds on their Rademacher complexity. Instead of using semidefinite programming we propose a simple, fast and efficient gradient-descent algorithm.

In section 2 we calculate the complexity of different classes of kernels. This yields a convex optimization problem. In section 3 we propose to restrict the optimization of the spectrum such that the order of the eigenvalues is preserved. This convex constraint is implemented by using polynomials of the kernel matrix with non–negative coefficients only.

In section 4 we use gradient descent to implement the optimization algorithm. Experimental results on standard data sets (UCI Machine Learning Repository) show in section 5 that indeed overfitting happens if we do not keep the order of the eigenvalues.

## 2   Bounding the Rademacher Complexity of Matrix Classes

Let us introduce some notation. Let $\mathcal{X}$ be a measurable space (the instance space) and $\mathcal{Y} = \{-1, 1\}$. We consider here the setting of transduction where the data is generated as follows. A fixed sample of size $2n$ is given $(x_1, y_1), \ldots, (x_{2n}, y_{2n})$ and a permutation $\pi$ of $\{1, \ldots, 2n\}$ is chosen at random (uniformly). The algorithm is given $X_1 = x_{\pi(1)}, \ldots, X_{2n} = x_{\pi(2n)}$ and $Y_1 = y_{\pi(1)}, \ldots, Y_n = y_{\pi(n)}$, i.e. it has access to all instances but to the labels of the first $n$ instances only.

The algorithm picks some classifier $f : \mathcal{X} \to \mathbb{R}$ and the goal is to minimize the error of this classifier on the test instances. Let us denote by $L(f)$ the error of $f$ on the testing instances, $L(f) := \frac{1}{n} \sum_{i=n+1}^{2n} I(Y_i f(X_i) \leq 0)$. The empirical Rademacher complexity of a set $\mathcal{F}$ of functions from $\mathcal{X}$ to $\mathbb{R}$ is defined as

$$R(\mathcal{F}) := \mathbb{E}_\sigma \left[ \sup_{f \in \mathcal{F}} \frac{1}{n} \sum_{i=1}^{2n} \sigma_i f(X_i) \right],$$

where the expectation is taken with respect to the independent Rademacher random variables $\sigma_i$ ($P(\sigma_i = 1) = P(\sigma_i = -1) = 1/2$). For a vector $v$, $v \geq 0$ means that all the components of $v$ are non-negative. For a matrix $K$, $K \geq 0$ means that $K$ is positive definite.

## 2.1 General Bound

We denote by $\phi$ the function defined as $\phi(x) = 1$ for $x \leq 0$, $\phi(x) = 0$ for $x > 1$ and $\phi(x) = 1 - x$ otherwise. From the proof of Theorem 1 in [5] we obtain the lemma below.

**Lemma 1** *Let $\mathcal{F}$ be a set of real-valued functions. For any $\delta > 0$, with probability at least $1 - \delta$, for all $f \in \mathcal{F}$ we have*

$$L(f) \leq \frac{1}{n}\sum_{i=1}^{n}\phi(Y_i f(X_i)) + \mathbb{E}_\sigma\left[\sup_{f\in\mathcal{F}}\frac{1}{n}\sum_{i=1}^{2n}\sigma_i\phi(Y_i f(X_i))\right] + \sqrt{\frac{\log 1/\delta}{n}} + \frac{4}{\sqrt{n}}\ .$$

Using the comparison inequality for Rademacher processes in [6] we immediately obtain the following corollary.

**Corollary 1** *Let $\mathcal{F}$ be a set of real-valued functions. For any $\delta > 0$, with probability at least $1 - \delta$, for all $f \in \mathcal{F}$ we have*

$$L(f) \leq \frac{1}{n}\sum_{i=1}^{n}\phi(Y_i f(X_i)) + R(\mathcal{F}) + \sqrt{\frac{\log 1/\delta}{n}} + \frac{4}{\sqrt{n}}\ .$$

Now we will apply this bound to several different classes of functions. We will thus compute $R(\mathcal{F})$ for each of those classes. For a positive definite kernel $k : \mathcal{X} \times \mathcal{X} \to \mathbb{R}$ one considers usually the RKHS formed by the closure of $\mathrm{span}\{k(x,.) : x \in \mathcal{X}\}$ with respect to the inner product defined by $\langle k(x,.), k(x',.)\rangle = k(x,x')$. Since we will vary the kernel it is convenient to distinguish between the vectors in the RKHS and their geometric relation. We first define the abstract real vectors space $\mathcal{G} = \mathrm{span}\{\delta_x : x \in \mathcal{X}\}$ where $\delta_x$ is the evaluation functional at point $x$. Then we define for a given kernel $k$ the Hilbert space $\mathcal{G}(k)$ as the closure of $\mathcal{G}$ with respect to the scalar product given by $\langle \delta_x, \delta_{x'}\rangle = k(x,x')$. In this way we can vary $k$, i.e. the geometry, without changing the vector space structure of any finite dimensional subspace of the form $\mathrm{span}\{\delta_{x_1},\ldots,\delta_{x_n}\}$. We can identify the RKHS above with $\mathcal{G}(k)$ via $\delta_x = k(x,\cdot)$ and $\langle \delta_x, \delta_{x'}\rangle = \langle k(x,\cdot), k(x',\cdot)\rangle$.

**Lemma 2** *Let $k$ be a kernel on $\mathcal{X}$, let $x_1,\ldots,x_n \in \mathcal{X}$ and $K = (k(X_i,X_j))_{ij}$. For all $M > 0$ we have*

$$\mathbb{E}_\sigma\left[\sup_{g\in\mathcal{G}(k),\|g\|\leq 1/M}\sum_{i=1}^{n}\sigma_i\langle g,\delta_{X_i}\rangle\right] = \frac{1}{M}\mathbb{E}_\sigma\left[\sqrt{\langle\sigma,K\sigma\rangle}\right]\ .$$

**Proof:** We have

$$\sup_{\|g\|\leq 1/M}\sum_{i=1}^{n}\sigma_i\langle g,\delta_{X_i}\rangle = \sup_{\|g\|\leq 1/M}\left\langle g,\sum_{i=1}^{n}\sigma_i\delta_{X_i}\right\rangle = \frac{1}{M}\left\|\sum_{i=1}^{n}\sigma_i\delta_{X_i}\right\| = \frac{1}{M}\sqrt{\langle\sigma,K\sigma\rangle}\ .$$

The second equality holds due to Cauchy–Schwarz inequality which becomes here an equality because of the supremum. Notice that $\langle\sigma,K\sigma\rangle$ is always non–negative since $K$ is positive definite. Taking expectations concludes the proof. $\square$

The expression in lemma 2 is up to a factor $1/n$ the Rademacher complexity of the class of functions in $\mathcal{G}(k)$ with margin $M$. It is important to notice that this is equal to the Rademacher complexity of the subspace of $\mathcal{G}(k)$ which is spanned by the data. Indeed, let us consider the space $\mathcal{G}_n := \mathrm{span}\{\delta_{x_1},\ldots,\delta_{x_n}\}$ then this is a Hilbert subspace of $\mathcal{G}(k)$. Moreover, we have

$$\sup_{\|g\|\leq 1/M, g\in\mathcal{G}_n}\sum_{i=1}^{n}\sigma_i\langle g,\delta_{X_i}\rangle = \frac{1}{M}\left\|\sum_{i=1}^{n}\sigma_i\delta_{X_i}\right\|_{\mathcal{G}_n} = \frac{1}{M}\sqrt{\langle\sigma,K\sigma\rangle}\ .$$

This proves that we are actually capturing the right complexity since we are computing the complexity of the set of hyperplanes whose normal vector can be expressed as a linear combination of the data points. Now, let's assume that we allow the kernel to change, that is, we have a set of possible kernels $\mathcal{K}$, or equivalently a set of possible kernel matrices. Let $\mathcal{G}_n(K)$ be $\mathcal{G}_n$ with the inner product induced by $K$ and let $\mathcal{F}_K$ denote the class of hyperplanes with margin $M$ in the space $\mathcal{G}_n(K)$, and $\mathcal{F}_{\mathcal{K}} := \cup_{K \in \mathcal{K}} \mathcal{F}_K$. Using lemma 2 we have

$$R(\mathcal{F}_{\mathcal{K}}) = \mathbb{E}_\sigma \left[ \sup_{\substack{K \in \mathcal{K}, g \in \mathcal{G}_n(K) \\ \|g\| \leq 1/M}} \frac{1}{n} \sum_{i=1}^{2n} \sigma_i \langle g, \delta_{X_i} \rangle \right] = \frac{1}{nM} \mathbb{E}_\sigma \left[ \sup_{K \in \mathcal{K}} \sqrt{\langle \sigma, K\sigma \rangle} \right] . \qquad (1)$$

Let $\|K\|_2$ denote the Frobenius norm of $K$, i.e. $\|K\|_2^2 = \sum_{i,j} K_{i,j}^2$. Recall that for symmetric positive definite matrices, the Frobenius norm is equal to the 2-norm of the spectrum, i.e. $\|K\|_2^2 = \sum_{i=1}^n \lambda_i^2$. Also, recall that the trace of such a matrix is equal to the 1-norm of its spectrum, i.e. $\operatorname{tr} K = \sum_{i=1}^n K_{i,i} = \sum_{i=1}^n \lambda_i$. Finally, recall that for a positive definite matrix $K$ the operator norm $\|K\|$ is given by $\|K\| = \sup_{\|v\| \leq 1} v^T K v = \max\{\lambda_1, \ldots, \lambda_n\}$. We will denote $\|K\|_\infty := \|K\|$ and $\|K\|_1 := \operatorname{tr} K$.

It is easy to see that for a fixed kernel matrix $K$, we have $R(\mathcal{F}_K) \leq \sqrt{\operatorname{tr} K}/Mn$. Also, it is useful to keep in mind that for certain kernels like the RBF kernel, the trace of the kernel matrix grows approximately linearly in the number of examples, while even if the problem is linearly separable, the margin decreases in the best case to a fixed strictly positive constant. This means that we have $\sqrt{\operatorname{tr} K}/Mn = O(1/\sqrt{n})$.

## 2.2 Complexity of $p$-balls of kernel matrices

The first class that one may consider is the class of all positive definite matrices with $p$-norm bounded by some constant.

**Theorem 1** *Let $c > 0$ and $p \in \{1, 2, \infty\}$. Define $\mathcal{K}_p = \{K \geq 0 : \|K\|_p \leq c\}$, then*

$$R(\mathcal{F}_{K_p}) = \frac{1}{Mn} \sqrt{2cn} .$$

**Proof:** Using (1) we thus have to compute $\mathbb{E}_\sigma \left[ \sup_{K \in \mathcal{K}_p} \sqrt{\langle \sigma, K\sigma \rangle} \right]$. Since we can always find some $K \in \mathcal{K}_p$ having an eigenvector $\sigma$ with eigenvalue $c$ we obtain $\sup_{K \in \mathcal{K}_p} \sqrt{\langle \sigma, K\sigma \rangle} = \sqrt{c} \|\sigma\| = \sqrt{2cn}$. which concludes the proof. $\qquad \square$

*Remark*: Observe that $\mathcal{K}_1 \subsetneq \mathcal{K}_2 \subsetneq \mathcal{K}_\infty$ for the same value of $c$. However they have the same Rademacher complexity. From the proof we see that for the calculation of the complexity only the *contribution* of $K$ in direction of $\sigma$ matters. Therefore for every $\sigma$ the *worst case* element is contained in all three classes.

Recall that in the case of the RBF kernel we have $\sqrt{c}/Mn = O(1/\sqrt{n})$ which means that we would obtain in this case a Rademacher complexity which does not decrease with $n$. It seems clear that proper learning is not possible in such a class, at least from the view point of this way of measuring the complexity.

## 2.3 Complexity of the convex hull of kernel matrices

Lanckriet et al. [5] considered positive definite linear combinations of $\ell$ kernel matrices, i.e. the class

$$\mathcal{K} = \{ K = \sum_{i=1}^\ell d_i K_i : \operatorname{tr} K = c, K \geq 0 \} . \qquad (2)$$

We rather consider the (smaller) class

$$\mathcal{K} = \{K = \sum_{i=1}^{\ell} d_i K_i : \operatorname{tr} K = c, d \geq 0\}. \tag{3}$$

which has simple linear constraints on the feasible parameter set and allows us to use a straightforward gradient descent algorithm. Notice that $\mathcal{K}$ is the convex hull of the matrices $\tilde{K}_1, \ldots, \tilde{K}_\ell$ where $\tilde{K}_i = cK_i / \operatorname{tr} K_i$.

We obtain the following bound on the Rademacher complexity of this class.

**Theorem 2** *Let $K_1, \ldots, K_\ell$ be some fixed kernel matrices and $\mathcal{K}$ as defined in (3) then*

$$R(\mathcal{F}_\mathcal{K}) \leq \frac{1}{M} \sqrt{\frac{2c}{n}} \sqrt{\max_{i=1,\ldots,\ell} \frac{\|K_i\|}{\operatorname{tr} K_i}}$$

**Proof:** Applying Jensen inequality to equation (1) we calculate first

$$\sup_{K \in \mathcal{K}} \sum_{i=1}^{\ell} d_i \langle \sigma, K_i \sigma \rangle = c \max_{i=1,\ldots,\ell} \frac{\langle \sigma, K_i \sigma \rangle}{\operatorname{tr} K_i} \leq 2cn \max_{i=1,\ldots,\ell} \frac{\|K_i\|}{\operatorname{tr} K_i}.$$

Indeed, consider the sum as a dot product and identify the domain of $d$. Then one recognizes that the first equality holds since the supremum is obtained for $d$ at one of the vectors $(0, \ldots, 0, c/\operatorname{tr} K_i, 0, \ldots, 0)$. The second part is due to the fact $\langle \sigma, K_i \sigma \rangle \leq \|K_i\| \|\sigma\|^2 = 2n\|K_i\|$. $\square$

*Remark:* For a large class of kernel functions the trace of the induced kernel matrix scales linearly in the sample size $n$. Therefore we have to scale $c$ linearly with $n$. On the other hand the operator norm of the induced kernel matrix grows sublinearly in $n$. If the margin is bounded we can therefore ensure learning. With other words, if the kernels inducing $K_1, \ldots, K_n$ are consistent, then the convex hull of the kernels is also consistent.

*Remark:* The bound on the complexity for this class is less then the one obtained by Lanckriet et al. [5] for their class. Furthermore, it contains only easily computable quantities. Recognize that in the proof of the above theorem there appears a quantity similar to the maximal alignment of a kernel to arbitrary labels. It is interesting to notice also that the Rademacher complexity somehow measures the average alignment of a kernel to random labels.

## 2.4 Complexity of spectral classes of kernels

Although the class defined in (3) has smaller complexity than the one in (2), we may want to restrict it further. One way of doing so is to consider a set of matrices which have the same eigenvectors. Generally speaking, the kernel encodes some prior about the data and we may want to retain part of this prior and allow the rest to be tuned from the data. A kernel matrix can be decomposed into two parts: its set of eigenvectors and its spectrum (set of eigenvalues). We will fix the eigenvectors and tune the spectrum from the data.

For a kernel matrix $K_0 = UDU^t$ and $c > 0$ we consider the spectral class of $K_0$, given by

$$\mathcal{K} = \{K : \operatorname{tr} K = c, U^t K U \text{ is diag.}\} = \{f(K_0) : \operatorname{tr} f(K_0) = c, f \in C(\mathbb{R}_0^+)\} \tag{4}$$

Notice that this class can be considered as the convex hull of the matrices $cv_i v_i^t$ where $v_i$ are the eigenvectors (columns of $U$).

*Remark:* We assume that all eigenvalues are different, otherwise the above sets do not agree. Note that Cristianini et al. proposed to optimize the alignment over this class.

We obtain the following bound on the complexity of such a class.

**Theorem 3** *Let $c \geq 0$, let $U$ be some fixed unitary matrix and $\mathcal{K}$ as defined in* (4), *then for all $M > 0$*

$$R(\mathcal{F}_K) \leq \frac{1}{nM}\sqrt{2c \ln 2n}\,.$$

**Proof:** As before we start with Equation (1). If we denote $v = U^t \sigma$ and obtain

$$\mathbb{E}_\sigma \left[ \sup_{K \in \mathcal{K}} \sqrt{\sum_{i=1}^n \lambda_i v_i^2} \right] \leq \sqrt{c}\,\mathbb{E}_\sigma \left[ \sqrt{\max_{i=1,\ldots,n} v_i^2} \right] = \sqrt{c}\,\mathbb{E}_\sigma \left[ \max_{i=1,\ldots,n} |v_i| \right]\,.$$

Note that $v_i = \sum_{j=1}^n u_{ij}\sigma_j$ so that, using Lemma 2.2 in [3] and the fact that $\sum_{i=1}^n u_{ij}^2 = 1$, we obtain the result. $\qquad\square$

*Remark:* As a corollary, we obtain that for any number $\ell$ of kernel matrices $K_1, \ldots, K_\ell$ which *commute*, the same bound holds on the complexity of their convex hull.

## 3 Optimizing the Kernel

In order to choose the right kernel, we will now consider the bound of Corollary 1. For a fixed kernel, the complexity term in this bound is proportional to $\sqrt{\operatorname{tr} K}/Mn$. We will consider a class of kernels and pick the one that minimizes this bound. This suggests to keep the trace fixed and to maximize the margin.

Using Corollary 1 with the bounds derived in Section 2 we immediately obtain a generalization bound for such a procedure.

Theorem 3 suggests that optimizing the whole spectrum of the kernel matrix does not significantly increase the complexity. However experiments (see Section 5) show that overfitting occurs. We present here a possible explanation for this phenomenon.

Loosely speaking, the kernel encodes some prior information about how the labels two data points should be coupled. Most often this prior corresponds to the knowledge that two similar data points should have a similar label.

Now, when optimizing over the spectrum of a kernel matrix, we replace the prior of the kernel function by information given by the data points. It turns out that this leads to overfitting in practical experiments. In section 2.4 we have shown that the complexity of the spectral class is not significantly bigger than the complexity for a fixed kernel, thus the complexity is not a sufficient explanation for this phenomenon.

It is likely that when optimizing the spectrum, some crucial part of the prior knowledge is lost. To verify this assumption, we ran some experiments on the real line. We have to separate two clouds of points in $\mathbb{R}$. When the clouds are well separated, a Gaussian kernel easily deals with the task while if we optimize the spectrum of this kernel with respect to the margin criterion, the classification has arbitrary jumps in the middle of the clouds.

A possible way of retaining more of the spatial information contained in the kernel is to keep the order of the eigenvalues fixed. It turns out that in the same experiments, when the eigenvalues are optimized keeping their original order, no spurious jumps occur.

We thus propose to add the extra constraint of keeping the order of the eigenvalues fix. This constrain is fulfilled by restricting the functions in (4) to polynomials of degree $\ell \in \{1, \ldots, n\}$ with non–negative coefficients, i.e. we consider spectral optimization by convex, non–decreasing functions. For a given kernel matrix $K_0$, we thus define

$$\mathcal{K} = \{K = \sum_{s=1}^\ell d_i K_0^i : \operatorname{tr} K = c,\, d \geq 0\}\,. \tag{5}$$

Indeed, recent results shows that the Rademacher complexity is reduced in this way [7].

## 4 Implementation

Following Lanckriet et al. [5] one can formulate the problem of optimizing the margin error bound optimization as a semidefinite programming problem. Here we considered classes of kernels that can be written as linear combinations of kernel matrices with non-negative coefficients and fixed trace. In that case, one obtains the following problem (the subscript $tr$ indicates that we keep the block corresponding to the training data only)

$$\min_{d,t,\lambda,\nu} t \quad \text{subject to} \quad \sum_{i=1}^{\ell} d_i \operatorname{tr} K_i = c, \quad \nu \geq 0, \quad d \geq 0,$$

$$\begin{pmatrix} Y(\sum_{i=1}^{\ell})_{tr} Y & e + \nu + \lambda y \\ (e + \nu + \lambda y)^t & t \end{pmatrix} \geq 0,$$

It turns out that implementing this semidefinite program is computationally quite expensive. We thus propose a different approach based on the work of [1]. Indeed, the goal is to minimize a bound of the form $\frac{\operatorname{tr} K}{M^2}$ so that if we fix the trace, we simply have to minimize the squared norm of the solution vector $w$. It has been proven in [1] that the gradient of $\|w\|^2$ can be computed as

$$\frac{\partial \|w\|^2}{\partial \theta} = -\alpha^T Y \frac{\partial K}{\partial \theta} \alpha . \tag{6}$$

The algorithm we suggest can thus be described as follows

1. Train an SVM to find the optimal value of $\alpha$ with the current kernel matrix.
2. Make a gradient step according to (6). Here, $\frac{\partial \|w\|^2}{\partial d_i} = -\alpha^T Y (K_i)_{tr} Y \alpha$.
3. Enforce the constraints on the coefficients (normalization and non-negativity).
4. Return to 1 unless a termination criterion is reached.

It turns out that this algorithm is very efficient and much simpler to implement than semidefinite programming. Moreover, the semidefinite programming formulations involve a large amount of (redundant) variables, so that a typical SDP solver will take 10 to 100 times longer to perform the same task since it will not use the specific symmetries of the problem.

## 5 Experiments

In order to compare our results we use the same setting as in [5]: we consider the Breast cancer and Sonar databases from the UCI repository and perform 30 random splits with 60% of the data for training and 40% for testing. $K_1$ denotes the matrix induced by the polynomial kernel $k_1(x, y) = (1 + x \cdot y)^d$, $K_2$ the matrix induced by the Gaussian kernel $k_2(x, y) = \exp(-\|x - y\|^2 / 2\sigma)$, and $K_3$ the matrix by the linear kernel $k_3(x, y) = x \cdot y$.

First we compare two classes of kernels, linear combinations defined by (2) and convex combination by (3). Figure 1 shows that optimizing the margin on both classes yields roughly the same performance while optimizing the alignment with the ideal kernel is worse. Furthermore, considering the class defined in (3) yields a large improvement on computational efficiency.

Next, we compare the optimization of the margin over the classes (3), (4) and (5) with degree 3 polynomials. Figure 1 indicates that tuning the full spectrum leads to overfitting while keeping the order of the eigenvalues gives reasonable performance (this performance is retained when the degree of the polynomial is increased).

| | $K_1$ | $K_2$ | $K_3$ | $K_L$ | $K_A$ | $K_C$ | $K_S$ | $K_P$ |
|---|---|---|---|---|---|---|---|---|
| Breast cancer | $d=2$ | $\sigma=.5$ | - | - | - | - | - | - |
| $\sqrt{\mathrm{tr}\,K_*}/(Mn)$ | 25.1 | 1.09 | - | 0.54 | 0.55 | **0.53** | 0.42 | **0.9** |
| test error (%) | 7.1 | 10.8 | - | 4.2 | 3.8 | **3.3** | 30.8 | **10.9** |
| Sonar | $d=2$ | $\sigma=.1$ | - | - | - | - | - | - |
| $\sqrt{\mathrm{tr}\,K}/(Mn)$ | 9.65 | 1.34 | 49.0 | 1.14 | 1.22 | **1.17** | 0.92 | **1.23** |
| test error (%) | 18.8 | 25.1 | 27.4 | 16.4 | 24.4 | **18.0** | 33.0 | **21.4** |

Figure 1: Performance of optimized kernels for different kernel classes and optimization procedures (methods proposed in the present paper are typeset in bold face). $K_1, K_2$ and $K_3$ indicate fixed kernels, see text. $K_L$ given by (2) and maximized margin, cf. [5]; $K_A$ given by (3) and maximized alignment with the ideal kernel cf. [2]; $K_C$ given by (3) and maximized margin; $K_S$ given by (4), i.e. whole spectral class of $K_2$ and maximized margin; $K_P$ given by (5) with $\ell = 3$, i.e. keeping the order of the eigenvalues in the spectral class and maximized margin. The performance of $K_P$ is much better than of $K_S$.

## 6 Conclusion

We have derived new bounds on the Rademacher complexity of classes of kernels. These bounds give guarantees for the generalization error when optimizing the margin over a function class induced by several kernel matrices. We propose a general methodology for implementing the optimization procedure for such classes which is simpler and faster than semidefinite programming while retaining the performance. Although the bound for spectral classes is quite tight, we encountered overfitting in the experiments. We overcome this problem by keeping the order of the eigenvalues fix. The motivation of this additional convex constraint is to maintain more information about the similarity measure.

The condition to fix the order of the eigenvalues is a new type of constraint. More work is needed to understand this constrain and its relation to the prior knowledge contained in the corresponding class of similarity measures. The complexity of such classes seems also to be much smaller. Therefore we will investigate the generalization behavior on different natural and artificial data sets in future work. Another direction for further investigation is to refine the bounds we obtained, using for instance local Rademacher complexities.

## References

[1] O. Chapelle, V. Vapnik, O. Bousquet, and S. Mukherjee. Choosing multiple parameters for support vector machines. *Machine Learning*, 46(1):131–159, 2002.

[2] N. Cristianini, J. Kandola, A. Elisseeff, and J. Shawe-Taylor. On optimizing kernel alignment. *Journal of Machine Learning Research*, 2002. To appear.

[3] L. Devroye and G. Lugosi. Combinatorial Methods in Density Estimation. Springer-Verlag, New York, 2000.

[4] J. Kandola, J. Shawe-Taylor and N. Cristianini. Optimizing Kernel Alignment over Combinations of Kernels. In *Int Conf Machine Learning*, 2002. In press.

[5] G. Lanckriet, N. Cristianini, P. Bartlett, L. El Ghaoui, and M.I. Jordan. Learning the kernel matrix with semidefinite programming. In *Int Conf Machine Learning*, 2002. In press.

[6] M. Ledoux and M. Talagrand. Probability in Banach Spaces. Springer-Verlag, 1991.

[7] O. Bousquet, and D. J. L. Herrmann. Towards Structered Kernel Maschines. *Work in Progress*.
